# Active Learning For Identifying Function Threshold Boundaries

**Brent Bryan**
Center for Automated Learning and Discovery
Carnegie Mellon University
Pittsburgh, PA 15213
bryanba@cs.cmu.edu

**Jeff Schneider**
Robotics Institute
Carnegie Mellon University
Pittsburgh, PA 15213
schneide@cs.cmu.edu

**Robert C. Nichol**
Institute of Cosmology and Gravitation
University of Portsmouth
Portsmouth, PO1 2EG, UK
bob.nichol@port.ac.uk

**Christopher J. Miller**
Observatorio Cerro Tololo
Observatorio de AURA en Chile
La Serena, Chile
cmiller@noao.edu

**Christopher R. Genovese**
Department of Statistics
Carnegie Mellon University
Pittsburgh, PA 15213
genovese@stat.cmu.edu

**Larry Wasserman**
Department of Statistics
Carnegie Mellon University
Pittsburgh, PA 15213
larry@stat.cmu.edu

## Abstract

We present an efficient algorithm to actively select queries for learning the boundaries separating a function domain into regions where the function is above and below a given threshold. We develop experiment selection methods based on entropy, misclassification rate, variance, and their combinations, and show how they perform on a number of data sets. We then show how these algorithms are used to determine simultaneously valid $1 - \alpha$ confidence intervals for seven cosmological parameters. Experimentation shows that the algorithm reduces the computation necessary for the parameter estimation problem by an order of magnitude.

## 1 Introduction

In many scientific and engineering problems where one is modeling some function over an experimental space, one is not necessarily interested in the precise value of the function over an entire region. Rather, one is curious about determining the set of points for which the function exceeds some particular value. Applications include determining the functional range of wireless networks [1], factory optimization analysis, and gaging the extent of environmental regions in geostatistics. In this paper, we use this idea to compute confidence intervals for a set of cosmological parameters that affect the shape of the temperature power spectrum of the Cosmic Microwave Background (CMB).

In one dimension, the threshold discovery problem is a root-finding problem where no

hints as to the location or number of solutions are given; several methods exist which can be used to solve this problem (e.g. bisection, Newton-Raphson). However, one dimensional algorithms cannot be easily extended to the multivariate case. In particular, the ideas of root bracketing and function transversal are not well defined [2]; given a particular bracket of a continuous surface, there will be an infinite number of solutions to the equation $f(\vec{x}) - t = 0$, since the solution in multiple dimensions is a set of surfaces, rather than a set of points.

Numerous active learning papers deal with similar problems in multiple dimensions. For instance, [1] presents a method for picking experiments to determine the localities of local extrema when the input space is discrete. Others have used a variety of techniques to reduce the uncertainty over the problem's entire domain to map out the function (e.g. [3], and [4]), or locate the optimal value (e.g. [5]).

We are interested in locating the subset of the input space wherein the function is above a given threshold. Algorithms that merely find a local optimum and search around it will not work in general, as there may be multiple disjoint regions above the threshold. While techniques that map out the entire surface of the underlying function will correctly identify those regions which are above a given threshold, we assert that methods can be developed that are more efficient at localizing a particular contour of the function. Intuitively, points on the function that are located far from the boundary are less interesting, regardless of their variance. In this paper, we make the following contributions to the literature:

- We present a method for choosing experiments that is more efficient than global variance minimization, as well as other heuristics, when one is solely interested in localizing a function contour.
- We show that this heuristic can be used in continuous valued input spaces, without defining *a priori* a set of possible experiments (e.g. imposing a grid).
- We use our function threshold detection method to determine $1-\alpha$ simultaneously valid confidence intervals of CMB parameters, making no assumptions about the model being fit and few assumptions about the data in general.

## 2 Algorithm

We begin by formalizing the problem. Assume that we are given a bounded sample space $S \subset \mathbb{R}^n$ and a scoring function: $f : S \to \mathbb{R}$, but possibly no data points ($\{s, f(s)\}, s \in S$). Given a threshold $t$, we want to find the set of points $S'$ where $f$ is equal to or above the threshold: $\{s \in S' | s \in S, f(s) \geq t\}$. If $f$ is invertible, then the solution is trivial. However, it is often the case that $f$ is not trivially invertible, such as the CMB model mentioned in §1. In these cases, we can discover $S'$ by modeling $S$ given some experiments. Thus, we wish to know how to choose experiments that help us determine $S'$ efficiently.

We assume that the cost to compute $f(s)$ given $s$ is significant. Thus, care should be taken when choosing the next experiment, as picking optimum points may reduce the runtime of the algorithm by orders of magnitude. Therefore, it is preferable to analyze current knowledge about the underlying function and select experiments which quickly refine the estimate of the function around the threshold of interest. There are several methods one could use to create a model of the data, notably some form of parametric regression. However, we chose to approximate the unknown boundary as a Gaussian Process (GP), as many forms of regression (e.g. linear) necessarily smooths the data, ignoring subtle features of the function that may become pronounced with more data. In particular, we use ordinary kriging, a form of GPs, which assumes that the semivariogram ($\mathcal{K}(\cdot, \cdot)$ is a linear function of the distance between samples [6]; this estimation procedure assumes the the sampled data are normal with mean equal to the true function and variance given by the sampling noise . The expected value of $\mathcal{K}(s_i, s_j)$ for $s_i, s_j \in S$, is can be written as

$$E[\mathcal{K}(s_i, s_j)] = \frac{k}{2} \Big[ \sum_{l=1}^{n} \alpha_l (s_{il} - s_{jl})^2 \Big]^{1/2} + c$$

where $k$ is a constant — known as the kriging parameter — which is an estimated limit on the first derivate of the function, $\alpha_l$ is a scaling factor for each dimension, and $c$ is the variance (e.g. experimental noise) of the sampled points. Since, the joint distribution of a finite set of sampled points for GPs is Gaussian, the predicted distribution of a query point $s_q$ given a known set $A$ is normal with mean and variance given by

$$\mu_{s_q} = \mu_A + \Sigma'_{Aq}\Sigma_{AA}^{-1}(y_A - \mu_A) \tag{1}$$

$$\sigma^2_{s_q} = \Sigma'_{Aq}\Sigma_{AA}^{-1}\Sigma_{Aq} \tag{2}$$

where $\Sigma_{Aq}$ denotes the column vector with the $i$th entry equal to $\mathcal{K}(s_i, s_q)$, $\Sigma_{AA}$ denotes the semivariance matrix between the elements of $A$ (the $ij$ element of $\Sigma_{AA}$ is $\mathcal{K}(s_i, s_j)$), $y_A$ denotes the column vector with the $i$th entry equal to $f(s_i)$, the true value of the function for each point in $A$, and $\mu_A$ is the mean of the $y_A$'s.

As given, prediction with GP requires $O(n^3)$ time, as an $n \times n$ linear system of equations must be solved. However, for many GPs — and ordinary kriging in particular — the correlation between two points decreases as a function of distance. Thus, the full GP model can be approximated well by a local GP, where only the $k$ nearest neighbors of the query point are used to compute the prediction value; this reduces the computation time to $O(k^3 \log(n))$ per prediction, since $O(\log(n))$ time is required to find the k-nearest neighbors using spatial indexing structures such as balanced kd-trees.

Since we have assumed that experimentation is expensive, it would be ideal to iteratively analyze the entire input space and pick the next experiment in such a manner that minimized the total number of experiments necessary. If the size of the parameter space ($|S|$) is finite, such an approach may be feasible. However, if $|S|$ is large or infinite, testing all points may be impractical. Instead of imposing some arbitrary structure on the possible experimental points (such as using a grid), our algorithm chooses candidate points uniformly at random from the input space, and then selects the candidate point with the highest score according to the metrics given in §2.1. This allows the input space to be fully explored (in expectation), and ensures that interesting regions of space that would have fallen between successive grid points are not missed; in §4 we show how imposing a grid upon the input space results in just such a situation. While the algorithm is unable to consider the entire space for each sampling iteration, over multiple iterations it does consider most of the space, resulting in the function boundaries being quickly localized, as can be seen in §3.

## 2.1 Choosing experiments from among candidates

Given a set of random input points, the algorithm evaluates each one and chooses the point with the highest score as the location for the next experiment. Below is the list of evaluation methods we considered.

**Random:** One of the candidate points is chosen uniformly at random. This method serves as a baseline for comparison,

**Probability of incorrect classification:** Since we are trying to map the boundary between points above and below a threshold, we consider choosing the point from our random sample which has the largest probability of being misclassified by our model. Using the distribution defined by Equations 1 and 2, the probability, $p$, that the point is above the given threshold can be computed. The point is predicted to be above the threshold if $p > 0.5$ and thus the expected misclassification probability is $\min(p, 1 - p)$.

**Entropy:** Instead of misclassification probability we can consider entropy: $-p\log_2(p) - (1 - p)\log_2(1 - p)$. Entropy is a monotonic function of the misclassification rate so these two will not choose different experiments. They are listed separately because they have different effects when mixed with other evaluations. Both entropy and misclassification

will choose points near the boundary. Unfortunately, they have the drawback that once they find a point near the boundary they continue to choose points near that location and will not explore the rest of the parameter space.

**Variance:** Both entropy and probability of incorrect classification suffer from a lack of incentive to explore the space. To rectify this problem, we consider the variance of each query point (given by Equation 2) as an evaluation metric. This metric is common in active learning methods whose goal is to map out an entire function. Since variance is related to the distance to nearest neighbors, this strategy chooses points that are far from areas currently searched, and hence will not get stuck at one boundary point. However, it is well known that such approaches tend to spend a large portion of their time on the edges of the parameter space and ultimately cover the space exhaustively [7].

**Information gain:** Information gain is a common myopic metric used in active learning. Information gain at the query point is the same as entropy in our case because all run experiments are assumed to have the same variance. Computing a full measure of information gain over the whole state space would provide an optimal 1-step experiment choice. In some discrete or linear problems this can be done, but it is intractable for continuous non-linear spaces. We believe the good performance of the evaluation metrics proposed below stems from their being heuristic proxies for global information gain or reduction in misclassification error.

**Products of metrics:** One way to rectify the problems of point policies that focus solely on points near the boundary or points with large variance regardless of their relevance to refining the predictive model, is to combine the two measures. Intuitively, doing this can mimic the idea of information gain; the entropy of a query point measures the classification uncertainty, while the variance is a good estimator of how much impact a new observation would have in this region, and thus what fraction the uncertainty would be reduced. [1] proposed scoring points based upon the product of their entropy and variance to identify the presence of local maxima and minima, a problem closely related to boundary detection. We shall also consider scoring points based upon the product of their probability of incorrect classification and variance. Note that while entropy and probability of incorrect classification are monotonically related, entropy times variance and probability of incorrect classification times variance are not.

**Straddle:** Using the same intuition as for products of heuristics, we define straddle heuristic, as $\text{straddle}(s_q) = 1.96\hat{\sigma}_q - \left| \hat{f}(s_q) - t \right|$, The straddle algorithm scores points highest that are both unknown and near the boundary. As such, the straddle algorithm prefers points near the threshold, but far from previous examples. The straddle score for a point may be negative, which indicates that the model currently estimates the probability that the point is on a boundary is less than five percent. Since the straddle heuristic relies on the variance estimate, it is also subject to oversampling edge positions.

## 3  Experiments

We now assess the accuracy with which our model reproduces a known function for the point policies just described. This is done by computing the fraction of test points in which the predictive model agrees with the true function about which side of the threshold the test points are on after some fixed number of experiments. This process is repeated several times to account for variations due to the random sampling of the input space.

The first model we consider is a 2D sinusoidal function given by

$$f(x,y) = \sin(10x) + \cos(4y) - \cos(3xy) \qquad x \in [0,1], \quad y \in [0,2],$$

with a boundary threshold of $t = 0$. This function and threshold were examined for the following reasons: 1) the target threshold winds through the plot giving ample length to

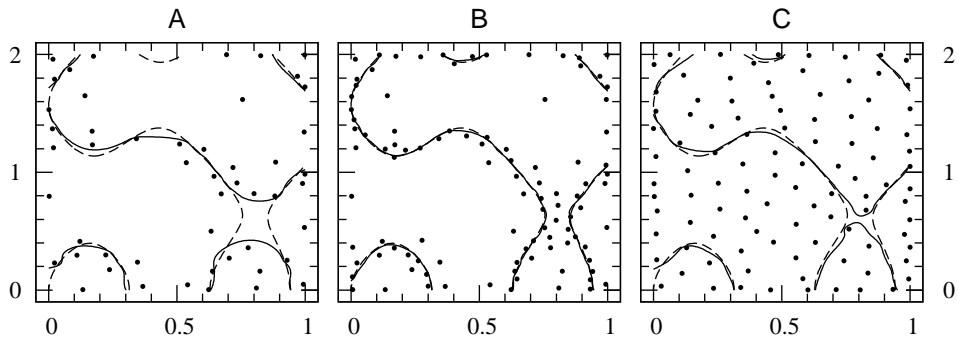

Figure 1: Predicted function boundary (solid), true function boundary (dashed), and experiments (dots) for the 2D sinusoid function after A) 50 experiments and B) 100 experiments using the straddle heuristic and C) 100 experiments using the variance heuristic.

Table 1: Number of experiments required to obtain 99% classification accuracy for the 2D models and 95% classification accuracy for the 4D model for various heuristics. Heuristics requiring more than 10,000 experiments to converge are labeled "did not converge".

|  | 2D Sin.(1K Cand.) | 2D Sin.(31 Cand.) | 2D DeBoor | 4D Sinusoid |
|---|---|---|---|---|
| Random | $617 \pm 158$ | $617 \pm 158$ | $7727 \pm 987$ | $6254 \pm 364$ |
| Entropy | did not converge | did not converge | did not converge | $6121 \pm 1740$ |
| Variance | $207 \pm 7$ | $229 \pm 9$ | $4306 \pm 573$ | $2320 \pm 57$ |
| Entropy$\times$Var | $117 \pm 5$ | $138 \pm 6$ | $1621 \pm 201$ | $1210 \pm 43$ |
| Prob. Incor.$\times$Std | $113 \pm 11$ | $129 \pm 14$ | $740 \pm 117$ | $1362 \pm 89$ |
| Straddle | $106 \pm 5$ | $123 \pm 6$ | $963 \pm 136$ | $1265 \pm 94$ |

test the accuracy of the approximating model, 2) the boundary is discontinuous with several small pieces, 3) there is an ambiguous region (around $(0.9, 1)$, where the true function is approximately equal to the threshold, and the gradient is small and 4) there are areas in the domain where the function is far from the threshold and hence we can ensure that the algorithm is not oversampling in these regions.

Table 1 shows the number of experiments necessary to reach a 99% and 95% accuracy for the 2D and 4D models, respectively. Note that picking points solely on entropy does not converge in many cases, while both the straddle algorithm and probability incorrect times standard deviation heuristic result in approximations that are significantly better than random and variance heuristics. Figures 1A-C confirm that the straddle heuristic is aiding in boundary prediction. Note that most of the 50 experiments sampled between Figures 1A and 1B are chosen near the boundary. The 100 experiments chosen to minimize the variance result in an even distribution over the input space and a worse boundary approximation, as seen in Figure 1C. These results indicate that the algorithm is correctly modeling the test function and choosing experiments that pinpoint the location of the boundary.

From the Equations 1 and 2, it is clear that the algorithm does not depend on data dimensionality directly. To ensure that heuristics are not exploiting some feature of the 2D input space, we consider the 4D sinusoidal function

$$f(\vec{x}) = \sin(10x_1) + \cos(4x_2) - \cos(3x_1x_2) + \cos(2x_3) + \cos(3x_4) - \sin(5x_3x_4)$$

where $\vec{x} \in [(0, 0, 1, 0), (1, 2, 2, 2)]$ and $t = 0$. Comparison of the 2D and 4D results in Table 1 reveals that the relative performance of the heuristics remains unchanged, indicating that the best heuristic for picking experiments is independent of the problem dimension.

To show that the decrease in the number candidate points relative to the input parameter space that occurs with higher dimensional problems is not an issue, we reconsider the 2D

sinusoidal problem. Now, we use only 31 candidate points instead of 1000 to simulate the point density difference between 4D and 2D. Results shown in Table 1, indicate that reducing the number of candidate points does not drastically alter the realized performance. Additional experiments were performed on a discontinuous 2D function (the DeBoor function given in [1]) with similar results, as can be seen in Table 1.

## 4  Statistical analysis of cosmological parameters

Let us now look at a concrete application of this work: a statistical analysis of cosmological parameters that affect formation and evolution of our universe. One key prediction of the Big Bang model for the origin of our universe is the presence of a $2.73K$ cosmic microwave background radiation (CMB). Recently, the Wilkinson Microwave Anisotropy Project (WMAP) has completed a detailed survey of the this radiation exhibiting small CMB temperature fluctuations over the sky [8]. It is believed that the size and spatial proximity of these temperature fluctuations depict the types and rates of particle interactions in the early universe and consequently characterize the formation of large scale structure (galaxies, clusters, walls and voids) in the current observable universe. It is conjectured that this radiation permeated through the universe unchanged since its formation 15 billion years ago. Therefore, the sizes and angular separations of these CMB fluctuations give an unique picture of the universe immediately after the Big Bang and have a large implication on our understanding of primordial cosmology.

An important summary of the temperature fluctuations is the CMB power spectrum shown in Figure 2, which gives the temperature variance of the CMB as a function of spatial frequency (or multi-pole moment). It is well known that the shape of this curve is affected by at least seven cosmological parameters: optical depth ($\tau$), dark energy mass fraction ($\Omega_\Lambda$), total mass fraction ($\Omega_m$), baryon density ($\omega_b$), dark matter density ($\omega_{dm}$), neutrino fraction ($f_n$), and spectral index ($n_s$). For instance, the height of first peak is determined by the total energy density of the universe, while the third peak is related to the amount of dark matter. Thus, by fitting models of the CMB power spectrum for given values of the seven parameters, we can determine how the parameters influence the shape of the model spectrum. By examining those models that fit the data, we can then establish the ranges of the parameters that result in models which fit the data.

Previous work characterizing confidence intervals for cosmological parameters either used marginalization over the other parameters, or made assumptions about the values of the parameters and/or the shape of the CMB power spectrum. However, [9] notes that "CMB data have now become so sensitive that the key issue in cosmological parameter determination is not always the accuracy with which the CMB power spectrum features can be measured, but often what prior information is used or assumed." In this analysis, we make no assumptions about the ranges or values of the parameters, and assume only that the data are normally distributed around the unknown CMB spectrum with covariance known up to a constant multiple. Using the method of [10], we create a non-parametric confidence ball (under a weighted squared-error loss) for the unknown spectrum that is centered on a nonparametric estimate with a radius for each specified confidence level derived from the asymptotic distribution of a pivot statistic[1]. For any candidate spectrum, membership in the confidence ball can be determined by comparing the ball's radius to the variance weighted sum of squares deviation between the candidate function and the center of the ball.

One advantage of this method is that it gives us simultaneously valid confidence intervals on all seven of our input parameters; this is not true for $1 - \alpha$ confidence intervals derived from a collection of $\chi^2$ distributions where the confidence intervals often have substantially lower coverage [11]. However, there is no way to invert the modeling process to determine parameter ranges given a fixed sum of squared error. Thus, we use the algorithm detailed

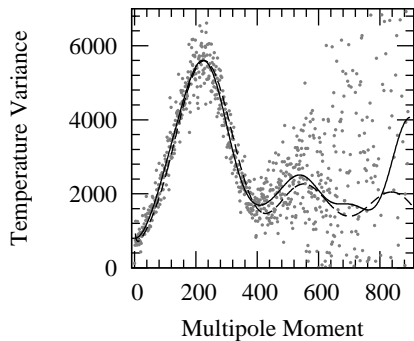

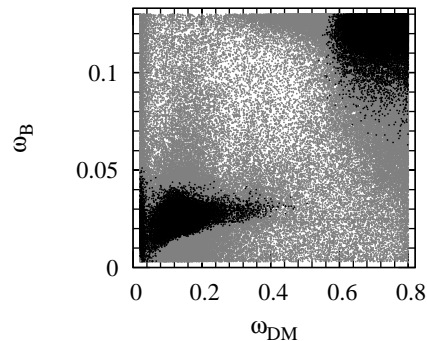

Figure 2: WMAP data, overlaid with regressed model (solid) and an example of a model CMB spectrum that barely fits at the 95% confidence level (dashed; parameter values are $\omega_{DM} = 0.1$ and $\omega_B = 0.028$).

Figure 3: 95% confidence bounds for $\omega_B$ as a function of $\omega_{DM}$. Gray dots denote models which are rejected at a 95% confidence level, while the black dots denote those that are not.

in §2 to map out the confidence surface as a function of the input parameters; that is, we use the algorithm to pick a location in the seven dimensional parameter space to perform an experiment, and then run CMBFast [12] to create simulated power spectrum given this set of input parameters. We can then compute the sum of squares of error for this spectrum (relative to the regressed model) and easily tell if the 7D input point is inside the confidence ball. In practice, we model the sum of squared error, not the confidence level of the model. This creates a more linear output space, as the confidence level for most of the models is zero, and thus it is impossible to distinguish between poor and terrible model fits.

Due to previous efforts on this project, we were able to estimate the semivariogram of the GP from several hundred thousand random points already run through CMBFast. For this work, we chose the $\alpha_l$'s such that the partials in each dimension where approximately unity, resulting in $k \simeq 1$; $c$ was set to a small constant to account for instabilities in the simulator. These points also gave a starting point for our algorithm[2]. Subsequently, we have run several hundred thousand more CMBFast models. We find that it takes 20 seconds to pick an experiment from among a set of 2,000 random candidates. CMBFast then takes roughly 3 minutes to compute the CMB spectrum given our chosen point in parameter space.

In Figure 3, we show a plot of baryon density ($\omega_B$) versus the dark matter density ($\omega_{DM}$) of the universe over all values of the other five parameters ($\tau, \Omega_{DE}, \Omega_M, f_n, n_s$). Experiments that are within a 95% confidence ball given the CMB data are plotted in black, while those that are rejected at the 95% level are gray. Note how there are areas that remain unsampled, while the boundary regions (transitions between gray and black points) are heavily sampled, indicating that our algorithm is choosing reasonable points. Moreover, the results of Figure 3 agree well with results in the literature (derived using parametric models and Bayesian analysis), as well as with predictions favored by nucleosynthesis [9].

While hard to distinguish in Figure 3, the bottom left group of points above the 95% confidence boundary splits into two separate peaks in parameter space. The one to the left is the concordance model, while the second peak (the one to the right) is not believed to represent the correct values of the parameters (due to constraints from other data). The existence of high probability points in this region of the parameter space has been suggested before, but computational limitations have prevented much characterization of it. Moreover, the third peak, near the top right corner of Figure 3 was basically ignored by previous grid based approaches. Comparison of the number of experiments performed by our straddle

Table 2: Number of points found in the three peaks for the grid based approach of [9] and our straddle algorithm.

| | Peak Center | | # Points in Effective Radius | |
| --- | --- | --- | --- | --- |
| | $\omega_{\mathrm{DM}}$ | $\omega_{\mathrm{B}}$ | Grid | Straddle |
| Concordance Model | 0.116 | 0.024 | 2118 | 16055 |
| Peak 2 | 0.165 | 0.023 | 2825 | 9634 |
| Peak 3 | 0.665 | 0.122 | 0 | 5488 |
| Total Points | | | 5613300 | 603384 |

algorithm with the grid based approach used by [9] is shown in Table 2. Even with only 10% of the experiments used in the grid approach, we sampled the concordance peak 8 times more frequently, and the second peak 3.4 times more frequently than the grid based approach. Moreover, it appears that the grid completely missed the third peak, while our method sampled it over 5000 times. These results dramatically illustrate the power of our adaptive method, and show how it does not suffer from assumptions made by a grid-based approaches. We are following up on the scientific ramifications of these results in a separate astrophysics paper.

## 5 Conclusions

We have developed an algorithm for locating a specified contour of a function while minimizing the number queries necessary. We described and showed how several different methods for picking the next experimental point from a group of candidates perform on synthetic test functions. Our experiments indicate that the straddle algorithm outperforms previously published methods, and even handles functions with large discontinuities. Moreover, the algorithm is shown to work on multi-dimensional data, correctly classifying the boundary at a 99% level with half the points required for variance minimizing methods. We have then applied this algorithm to a seven dimensional statistical analysis of cosmological parameters affecting the Cosmic Microwave Background. With only a few hundred thousand simulations we are able to accurately describe the interdependence of the cosmological parameters, leading to a better understanding of fundamental physical properties.

## Footnotes

[1]See Appendix 3 in [10] for the derivation of this radius

[2]While initial values are not required (as we have seen in §3), it is possible to incorporate this background knowledge into the model to help the algorithm converge more quickly.

## References

[1] N. Ramakrishnan, C. Bailey-Kellogg, S. Tadepalli, and V. N. Pandey. Gaussian processes for active data mining of spatial aggregates. In *Proceedings of the SIAM International Conference on Data Mining*, 2005.

[2] W. H. Press, S. A. Teukolsky, W. T. Vetterling, and B. P. Flannery. *Numerical Recipes in C*. Cambridge University Press, 2nd edition, 1992.

[3] D. A. Cohn, Z. Ghahramani, and M. I. Jordan. Active learning with statistical models. In G. Tesauro, D. Touretzky, and T. Leen, editors, *Advances in Neural Information Processing Systems*, volume 7, pages 705–712. The MIT Press, 1995.

[4] Simon Tong and Daphne Koller. Active learning for parameter estimation in bayesian networks. In *NIPS*, pages 647–653, 2000.

[5] A. Moore and J. Schneider. Memory-based stochastic optimization. In D. Touretzky, M. Mozer, and M. Hasselm, editors, *Neural Information Processing Systems 8*, volume 8, pages 1066–1072. MIT Press, 1996.

[6] Noel A. C. Cressie. *Statistics for Spatial Data*. Wiley, New York, 1991.

[7] D. MacKay. Information-based objective functions for active data selection. *Neural Computation*, 4(4):590–604, 1992.

[8] C. L. Bennett et al. First-Year Wilkinson Microwave Anisotropy Probe (WMAP) Observations: Preliminary Maps and Basic Results. *Astrophysical Journal Supplement Series*, 148:1–27, September 2003.

[9] M. Tegmark, M. Zaldarriaga, and A. J. Hamilton. Towards a refined cosmic concordance model: Joint 11-parameter constraints from the cosmic microwave background and large-scale structure. *Physical Review D*, 63(4), February 2001.

[10] C. Genovese, C. J. Miller, R. C. Nichol, M. Arjunwadkar, and L. Wasserman. Nonparametric inference for the cosmic microwave background. *Statistic Science*, 19(2):308–321, 2004.

[11] C. J. Miller, R. C. Nichol, C. Genovese, and L. Wasserman. A non-parametric analysis of the cmb power spectrum. *Bulletin of the American Astronomical Society*, 33:1358, December 2001.

[12] U. Seljak and M. Zaldarriaga. A Line-of-Sight Integration Approach to Cosmic Microwave Background Anisotropies. *Astrophyical Journal*, 469:437–+, October 1996.
